# Statistical Theory of Overtraining – Is Cross-Validation Asymptotically Effective?

**S. Amari, N. Murata, K.-R. Müller\***
Dept. of Math. Engineering and Inf. Physics, University of Tokyo
Hongo 7-3-1, Bunkyo-ku, Tokyo 113, Japan

**M. Finke**
Inst. f. Logik, University of Karlsruhe
76128 Karlsruhe, Germany

**H. Yang**
Lab. f. Inf. Representation, RIKEN,
Wakoshi, Saitama, 351-01, Japan

## Abstract

A statistical theory for overtraining is proposed. The analysis treats realizable stochastic neural networks, trained with Kullback-Leibler loss in the *asymptotic* case. It is shown that the asymptotic gain in the generalization error is small if we perform early stopping, even if we have access to the optimal stopping time. Considering cross-validation stopping we answer the question: In what ratio the examples should be divided into training and testing sets in order to obtain the optimum performance. In the non-asymptotic region cross-validated early stopping always decreases the generalization error. Our large scale simulations done on a CM5 are in nice agreement with our analytical findings.

## 1 Introduction

Training multilayer neural feed-forward networks, there is a folklore that the generalization error decreases in an early period of training, reaches the minimum and then increases as training goes on, while the training error monotonically decreases. Therefore, it is considered advantageous to stop training at an adequate time or to use regularizers (Hecht-Nielsen [1989], Hassoun [1995], Wang et al. [1994], Poggio and Girosi [1990], Moody [1992], LeCun et al. [1990] and others). To avoid overtraining, the following stopping rule has been proposed based on cross-validation:

Divide all the available examples into two disjoint sets. One set is used for training. The other set is used for testing such that the behavior of the trained network is evaluated by using the test examples and training is stopped at the point that minimizes the testing error.

The present paper gives a mathematical analysis of the so-called overtraining phenomena to elucidate the folklore. We analyze the asymptotic case where the number $t$ of examples are very large. Our analysis treats 1) a realizable stochastic machine, 2) Kullback-Leibler loss (negative of the log likelihood loss), 3) asymptotic behavior where the number $t$ of examples is sufficiently large (compared with the number $m$ of parameters). We firstly show that asymptotically the gain of the generalization error is small even if we could find the optimal stopping time. We then answer the question: In what ratio, the examples should be divided into training and testing sets in order to obtain the optimum performance. We give a definite answer to this problem. When the number $m$ of network parameters is large, the best strategy is to use almost all $t$ examples in the training set and to use only $1/\sqrt{2m}$ examples in the testing set, e.g. when $m = 100$, this means that only 7% of the training patterns are to be used in the set determining the point for early stopping.

Our analytic results were confirmed by large-scale computer simulations of three-layer continuous feedforward networks where the number $m$ of modifiable parameters are $m = 100$. When $t > 30m$, the theory fits well with simulations, showing cross-validation is not necessary, because the generalization error becomes worse by using test examples to obtain an adaequate stopping time. For an intermediate range, where $t < 30m$ overtraining occurs surely and the cross-validation stopping improves the generalization ability strongly.

## 2  Stochastic feedforward networks

Let us consider a stochastic network which receives input vector $\mathbf{x}$ and emits output vector $\mathbf{y}$. The network includes a modifiable vector parameter $\mathbf{w} = (w_1, \cdots, w_m)$ and is denoted by $N(\mathbf{w})$. The input-output relation of the network $N(\mathbf{w})$ is specified by the conditional probability $p(\mathbf{y}|\mathbf{x}; \mathbf{w})$. We assume (a) that there exists a teacher network $N(\mathbf{w}_0)$ which generates training examples for the student $N(\mathbf{w})$. And (b) that the Fisher information matrix $G_{ij}(\mathbf{w}) = E\left[\frac{\partial}{\partial w_i} \log p(\mathbf{x}, \mathbf{y}; \mathbf{w}) \frac{\partial}{\partial w_j} \log p(\mathbf{x}, \mathbf{y}; \mathbf{w})\right]$ exists, is non-degenerate and is smooth in $\mathbf{w}$, where $E$ denotes the expectation with respect to $p(\mathbf{x}, \mathbf{y}; \mathbf{w}) = q(\mathbf{x})p(\mathbf{y}|\mathbf{x}; \mathbf{w})$. The training set $D_t = \{(\mathbf{x}_1, \mathbf{y}_1), \cdots, (\mathbf{x}_t, \mathbf{y}_t)\}$ consists of $t$ independent examples generated by the distribution $p(\mathbf{x}, \mathbf{y}; \mathbf{w}_0)$ of $N(\mathbf{w}_0)$. The maximum likelihood estimator (m.l.e.) $\hat{\mathbf{w}}$ is the one that maximizes the likelihood of producing $D_t$, or equivalently minimizes the training error or *empirical* risk function

$$R_{\text{train}}(\mathbf{w}) = -\frac{1}{t} \sum_{i=1}^{t} \log p(\mathbf{x}_i, \mathbf{y}_i; \mathbf{w}). \tag{2.1}$$

The generalization error or risk function $R(\mathbf{w})$ of network $N(\mathbf{w})$ is the expectation with respect to the true distribution,

$$R(\mathbf{w}) = -E_0[\log p(\mathbf{x}, \mathbf{y}; \mathbf{w})] = H_0 + D(\mathbf{w}_0 \parallel \mathbf{w}) = H_0 + E_0\left[\log \frac{p(\mathbf{x}, \mathbf{y}; \mathbf{w}_0)}{p(\mathbf{x}, \mathbf{y}; \mathbf{w})}\right], \tag{2.2}$$

where $E_0$ denotes the expectation with respect to $p(\mathbf{x}, \mathbf{y}; \mathbf{w}_0)$, $H_0$ is the entropy of the teacher network and $D(\mathbf{w}_0 \parallel \mathbf{w})$ is the Kullback-Leibler divergence from probability distribution $p(\mathbf{x}, \mathbf{y}; \mathbf{w}_0)$ to $p(\mathbf{x}, \mathbf{y}; \mathbf{w})$ or the divergence of $N(\mathbf{w})$ from $N(\mathbf{w}_0)$. Hence, minimizing $R(\mathbf{w})$ is equivalent to minimizing $D(\mathbf{w}_0 \parallel \mathbf{w})$, and the

minimum is attained at $\mathbf{w} = \mathbf{w}_0$. The asymptotic theory of statistics proves that the m.l.e. $\hat{\mathbf{w}}_t$ is asymptotically subject to the normal distribution with mean $\mathbf{w}_0$ and variance $G^{-1}/t$, where $G^{-1}$ is the inverse of the Fisher information matrix $G$. We can expand for example the risk $R(\mathbf{w}) = H_0 + \frac{1}{2}(\mathbf{w} - \mathbf{w}_0)^T G(\mathbf{w}_0)(\mathbf{w} - \mathbf{w}_0) + O\left(\frac{1}{t^2}\right)$ to obtain

$$\langle R_{\text{gen}}(\hat{\mathbf{w}}) \rangle = H_0 + \frac{m}{2t} + O\left(\frac{1}{t^2}\right), \quad \langle R_{\text{train}}(\hat{\mathbf{w}}) \rangle = H_0 - \frac{m}{2t} + O\left(\frac{1}{t^2}\right), \quad (2.3)$$

as asymptotic result for training and test error (see Murata et al. [1993] and Amari and Murata [1990]). An extension of (2.3) including higher order corrections was recently obtained by Müller et al. [1995].

Let us consider the gradient descent learning rule (Amari [1967], Rumelhart et al. [1986], and many others), where the parameter $\hat{\mathbf{w}}(n)$ at the $n$th step is modified by

$$\hat{\mathbf{w}}(n+1) = \hat{\mathbf{w}}(n) - \varepsilon \frac{\partial R_{\text{train}}(\hat{\mathbf{w}}_n)}{\partial \mathbf{w}}, \quad (2.4)$$

and where $\varepsilon$ is a small positive constant. This is batch learning where all the training examples are used for each iteration of modifying $\hat{\mathbf{w}}(n)$.[1] The batch process is deterministic and $\hat{\mathbf{w}}(n)$ converges to $\hat{\mathbf{w}}$, provided the initial $\mathbf{w}(0)$ is included in its basin of attraction. For large $n$ we can argue, that $\hat{\mathbf{w}}(n)$ is approaching $\hat{\mathbf{w}}$ isotropically and the learning trajectory follows a linear ray towards $\hat{\mathbf{w}}$ (for details see Amari et al. [1995]).

## 3  Virtual optimal stopping rule

During learning as the parameter $\hat{\mathbf{w}}(n)$ approaches $\hat{\mathbf{w}}$, the generalization behavior of network $N\{\hat{\mathbf{w}}(n)\}$ is evaluated by the sequence $R(n) = R\{\hat{\mathbf{w}}(n)\}$, $n = 1, 2, \ldots$ The folklore says that $R(n)$ decreases in an early period of learning but it increases later. Therefore, there exists an optimal stopping time $n$ at which $R(n)$ is minimized. The stopping time $n_{\text{opt}}$ is a random variable depending on $\hat{\mathbf{w}}$ and the initial $\mathbf{w}(0)$. We now evaluate the ensemble average of $\langle R(n_{\text{opt}}) \rangle$.

The true $\mathbf{w}_0$ and the m.l.e. $\hat{\mathbf{w}}$ are in general different, and they are apart of order $1/\sqrt{t}$. Let us compose a sphere $S$ of which the center is at $(1/2)(\mathbf{w}_0 + \hat{\mathbf{w}})$ and which passes through both $\mathbf{w}_0$ and $\hat{\mathbf{w}}$, as shown in Fig.1b. Its diameter is denoted by $d$, where $d^2 = |\hat{\mathbf{w}} - \mathbf{w}_0|^2$ and

$$E_0[d^2] = E_0[(\hat{\mathbf{w}} - \mathbf{w}_0)^T G^{-1}(\hat{\mathbf{w}} - \mathbf{w}_0)] = \frac{1}{t}\text{tr}(G^{-1}G) = \frac{m}{t}. \quad (3.1)$$

Let $A$ be the ray, that is the trajectory $\hat{\mathbf{w}}(n)$ starting at $\hat{\mathbf{w}}(0)$ which is not in the neighborhood of $\mathbf{w}_0$. The optimal stopping point $\mathbf{w}^*$ that minimizes

$$R(n) = H_0 + \frac{1}{2}|\hat{\mathbf{w}}(n) - \mathbf{w}_0|^2 \quad (3.2)$$

is given by the first intersection of the ray $A$ and the sphere $S$.

Since $\mathbf{w}^*$ is the point on $A$ such that $\mathbf{w}_0 - \mathbf{w}^*$ is orthogonal to $A$, it lies on the sphere $S$ (Fig.1b). When ray $A'$ is approaching $\hat{\mathbf{w}}$ from the opposite side of $\mathbf{w}_0$ (the right-hand side in the figure), the first intersection point is $\hat{\mathbf{w}}$ itself. In this case, the optimal stopping never occurs until it converges to $\hat{\mathbf{w}}$.

Let $\theta$ be the angle between the ray $A$ and the diameter $\mathbf{w}_0 - \hat{\mathbf{w}}$ of the sphere $S$. We now calculate the distribution of $\theta$ when the rays are isotropically distributed.

*Lemma 1.* When ray $A$ is approaching $\hat{\mathbf{w}}$ from the side in which $\mathbf{w}_0$ is included, the probability density of $\theta$, $0 \leq \theta \leq \pi/2$, is given by

$$r(\theta) = \frac{1}{I_{m-2}} \sin^{m-2} \theta, \quad \text{where} \quad I_m = \int_0^{\pi/2} \sin^m \theta d\theta. \tag{3.3}$$

The detailed proof of this lemma can be found in Amari et al. [1995]. Using the density of $\theta$ given by Eq.(3.3) and we arrive at the following theorem.

**Theorem 1**. The average generalization error at the optimal stopping point is given by

$$\langle R(n_{\text{opt}}) \rangle = H_0 + \frac{1}{2t}(m - \frac{1}{2}). \tag{3.4}$$

*Proof.* When ray $A$ is at angle $\theta$, $0 \leq \theta < \pi/2$, the optimal stopping point $\mathbf{w}^*$ is on the sphere $S$. It is easily shown that $|\mathbf{w}^* - \mathbf{w}_0| = d \sin \theta$. This is the case where $A$ is from the same side as $\mathbf{w}_0$ (from the left-hand side in Fig.1b), which occurs with probability 0.5, and the average of $(d \sin \theta)^2$ is

$$E_0[(d \sin \theta)^2] = \frac{E_0[d^2]}{I_{m-2}} \int_0^{\pi/2} \sin^2 \theta \sin^{m-2} \theta d\theta = \frac{m}{t} \frac{I_m}{I_{m-2}} = \frac{m}{t}(1 - \frac{1}{m}).$$

When $\theta$ is $\pi/2 \leq \theta \leq \pi$, that is $A$ approaches $\hat{\mathbf{w}}$ from the opposite side, it does not stop until it reaches $\hat{\mathbf{w}}$, so that $|\mathbf{w}^* - \mathbf{w}_0|^2 = |\hat{\mathbf{w}} - \mathbf{w}_0| = d^2$. This occurs with probability 0.5. Hence, we proved the theorem.

The theorem shows that, if we could know the optimal stopping time $n_{\text{opt}}$ for each trajectory, the generalization error decreases by $1/2t$, which has an effect of decreasing the effective dimensions by $1/2$. This effect is neglegible when $m$ is large. The optimal stopping time is of the order $\log t$. However, it is impossible to know the optimal stopping time. If we stop learning at an estimated optimal time $\hat{n}_{\text{opt}}$, we have a small gain when the ray $A$ is from the same side as $\mathbf{w}_0$ but we have some loss when ray $A$ is from the opposite direction. This shows that the gain is even smaller if we use a common stopping time $\bar{n}_{\text{opt}}$ independent of $\hat{\mathbf{w}}$ and $\mathbf{w}(0)$ as proposed by Wang et al. [1994]. However, the point is that there is neither direct means to estimate $n_{\text{opt}}$ nor $\bar{n}_{\text{opt}}$ rather than for example cross-validation. Hence, we analyze cross-validation stopping in the following.

## 4  Optimal stopping by cross-validation

The present section studies asymptotically two fundamental problems: 1) Given $t$ examples, how many examples should be used in the training set and how many in the testing set? 2) How much gain can one expect by the above cross-validated stopping?

Let us divide $t$ examples into $rt$ examples of the training set and $r't$ examples of the testing set, where $r + r' = 1$. Let $\hat{\mathbf{w}}$ be the m.l.e. from $rt$ training examples, and let $\tilde{\mathbf{w}}$ be the m.l.e. from the other $r't$ testing examples. Since the training examples and testing examples are independent, $\hat{\mathbf{w}}$ and $\tilde{\mathbf{w}}$ are subject to independent normal distributions with mean $\mathbf{w}_0$ and covariance matrices $G^{-1}/(rt)$ and $G^{-1}/(r't)$, respectively.

Let us compose the triangle with vertices $\mathbf{w}_0$, $\hat{\mathbf{w}}$ and $\tilde{\mathbf{w}}$. The trajectory $A$ starting at $\mathbf{w}(0)$ enters $\hat{\mathbf{w}}$ linearly in the neighborhood. The point $\mathbf{w}^*$ on the trajectory $A$ which minimizes the testing error is the point on $A$ that is closest to $\tilde{\mathbf{w}}$, since the testing error defined by

$$R_{\text{test}}(\mathbf{w}) = \frac{1}{r't} \sum_i \{-\log p(\mathbf{x}_i, \mathbf{y}_i; \mathbf{w})\}, \tag{4.1}$$

where summation is taken over $r't$ testing examples, can be expanded as

$$R_{\text{test}}(\mathbf{w}) = H_0 - \frac{1}{2}|\tilde{\mathbf{w}} - \mathbf{w}_0|^2 + \frac{1}{2}|\mathbf{w} - \tilde{\mathbf{w}}|^2. \qquad (4.2)$$

Let $S$ be the sphere centered at $(\hat{\mathbf{w}} + \tilde{\mathbf{w}})/2$ and passing through both $\hat{\mathbf{w}}$ and $\tilde{\mathbf{w}}$. It's diameter is given by $d = |\hat{\mathbf{w}} - \tilde{\mathbf{w}}|$. Then, the optimal stopping point $\mathbf{w}^*$ is given by the intersection of the trajectory $A$ and sphere $S$. When the trajectory comes from the opposite side of $\tilde{\mathbf{w}}$, it does not intersect $S$ until it converges to $\hat{\mathbf{w}}$, so that the optimal point is $\mathbf{w}^* = \hat{\mathbf{w}}$ in this case. Omitting the detailed proof, the generalization error of $\mathbf{w}^*$ is given by Eq.(??), so that we calculate the expectation

$$E[|\mathbf{w}^* - \mathbf{w}_0|^2] = \frac{m}{tr} - \frac{1}{2t}\left(\frac{1}{r} - \frac{1}{r'}\right).$$

*Lemma 2.* The average generalization error by the optimal cross-validated stopping is

$$\langle R(\mathbf{w}^*, r)\rangle = H_0 + \frac{2m-1}{4rt} + \frac{1}{4r't} \qquad (4.3)$$

We can then calculate the optimal division rate

$$r_{\text{opt}} = 1 - \frac{\sqrt{2m-1}-1}{2(m-1)} \quad \text{and} \quad r_{\text{opt}} = 1 - \frac{1}{\sqrt{2m}} \quad \text{(large $m$ limit).} \qquad (4.4)$$

of examples, which minimizes the generalization error. So for large $m$ only $(1/\sqrt{2m}) \times 100\%$ of examples should be used for testing and all others for training. For example, when $m = 100$, this shows that 93% of examples are to be used for training and only 7% are to be kept for testing. From Eq.(4.4) we obtain as optimal generalization error for large $m$

$$\langle R(\mathbf{w}^*, r_{\text{opt}})\rangle = H_0 + \frac{m}{2t}\left(1 + \sqrt{\frac{2}{m}}\right). \qquad (4.5)$$

This shows that the generalization error asymptotically *increases* slightly by cross-validation compared with non-stopped learning which is using *all* the examples for training.

## 5  Simulations

We use standard feed-forward classifier networks with $N$ inputs, $H$ sigmoid hidden units and $M$ softmax outputs (classes). The output activity $O_l$ of the $l$th output unit is calculated via the softmax squashing function

$$p(\mathbf{y} = C_l|\mathbf{x}; \mathbf{w}) = O_l = \frac{\exp(h_l)}{1 + \sum_k \exp(h_k)}, \quad l = 1, \cdots, M, \quad O_0 = \frac{1}{1 + \sum_k \exp(h_k)},$$

where $h_l^O = \sum_j w_{lj}^O s_j - \vartheta_l^O$ is the local field potential. Each output $O_l$ codes the a-posteriori probability of being in class $C_l$, $O_0$ denotes a zero class for normalization purposes. The $m$ network parameters consist of biases $\vartheta$ and weights $\mathbf{w}$. When $\mathbf{x}$ is input, the activity of the $j$-th hidden unit is

$$s_j = [1 + \exp(-\sum_{k=1}^{N} w_{jk}^H x_k - \vartheta_j^H)]^{-1}, \quad j = 1, \cdots, H.$$

The input layer is connected to the hidden layer via $\mathbf{w}^H$, the hidden layer is connected to the output layer via $\mathbf{w}^O$, but no short-cut connections are present. Although the network is completely deterministic, it is constructed to approximate

class conditional probabilities (Finke and Müller [1994]).

The examples $\{(\mathbf{x}_1, \mathbf{y}_1), \cdots, (\mathbf{x}_t, \mathbf{y}_t)\}$ are produced randomly, by drawing $\mathbf{x}_i$, $i = 1, \cdots, t$, from a uniform distribution independently and producing the labels $\mathbf{y}_i$ stochastically from the teacher classifier. Conjugate gradient learning with line-search on the empirical risk function Eq.(2.1) is applied, starting from some random initial vector. The generalization ability is measured using Eq. (2.2) on a large test set (50000 patterns). Note that we use Eq. (2.1) on the cross-validation set, because only the empirical risk is available on the cross-validation set in a practical situation. We compare the generalisation error for the settings: exhaustive training (no stopping), early stopping (controlled by the cross-validation set) and optimal stopping (controlled by the large testset). The simulations were performed on a parallel computer (CM5). Every curve in the figures takes about 8h of computing time on a 128 respectively 256 partition of the CM5, i.e. we perform 128-256 parallel trials. This setting enabled us to do extensive statistics (cf. Amari et al. [1995]). Fig. 1a shows the results of simulations, where $N = 8$, $H = 8$, $M = 4$, so that the number $m$ of modifiable parameters is $m = (N + 1)H + (H + 1)M = 108$. We observe clearly, that saturated learning without early stopping is the best in the asymptotic range of $t > 30m$, a range which is due to the limited size of the data sets often unaccessible in practical applications. Cross-validated early stopping does not improve the generalization error here, so that no overtraining is observed on the average in this range. In the asymptotic area (figure 1) we observe that the smaller the percentage of the training set, which is used to determine the point of early stopping, the better the performance of the generalization ability. When we use cross-validation, the optimal size of the test set is about 7% of all the examples, as the theory predicts.

Clearly, early stopping does improve the generalization ability to a large extent in an intermediate range for $t < 30m$ (see Müller et al. [1995]). Note, that our theory also gives a good estimate of the optimal size of the early stopping set in this intermediate range.

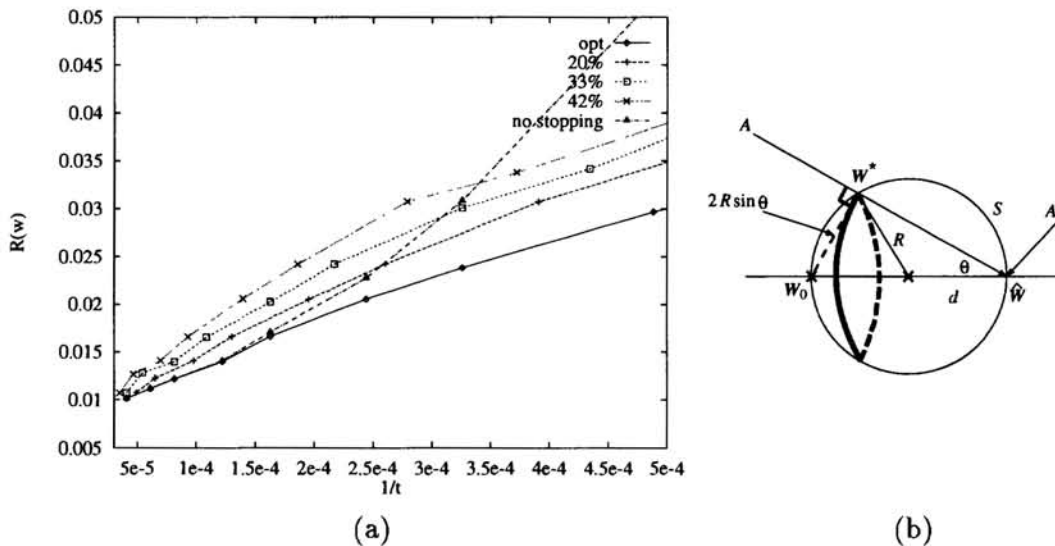

(a)                                              (b)

Figure 1: *(a) $R(\mathbf{w})$ plotted as a function of $1/t$ for different sizes $r'$ of the early stopping set for an 8-8-4 classifier network. opt. denotes the use of a very large cross-validation set (50000) and no stopping adresses the case where 100% of the training set is used for exhaustive learning. (b) Geometrical picture to determine the optimal stopping point $\mathbf{w}^*$.*

# 6   Conclusion

We proposed an asymptotic theory for overtraining. The analysis treats realizable stochastic neural networks, trained with Kullback-Leibler loss.

It is demonstrated both theoretically and in simulations that *asymptotically* the gain in the generalization error is small if we perform early stopping, even if we have access to the optimal stopping time. For cross-validation stopping we showed for large $m$ that optimally only $r'_{opt} = 1/\sqrt{2m}$ examples should be used to determine the point of early stopping in order to obtain the best performance. For example, if $m = 100$ this corresponds to using 93% of the $t$ training patterns for training and only 7% for testing where to stop. Yet, even if we use $r_{opt}$ for cross-validated stopping the generalization error is always increased comparing to exhaustive training. Nevertheless note, that this range is due to the limited size of the data sets often unaccessible in practical applications.

In the non-asymptotic region simulations show that cross-validated early stopping always helps to enhance the performance since it decreases the generalization error. In this intermediate range our theory also gives a good estimate of the optimal size of the early stopping set. In future we will consider higher order correction terms to extend our theory to give also a quantitative description of the non-asymptotic region.

**Acknowledgements:** We would like to thank Y. LeCun, S. Bös and K. Schulten for valuable discussions. K. -R. M. thanks K. Schulten for warm hospitality during his stay at the Beckman Inst. in Urbana, Illinois. We acknowledge computing time on the CM5 in Urbana (NCSA) and in Bonn, supported by the National Institutes of Health (P41RRO 5969) and the EC S & T fellowship (FTJ3-004, K. -R. M.).

## Footnotes

\*Permanent address: GMD FIRST, Rudower Chaussee 5, 12489 Berlin, Germany. E-mail: Klaus@first.gmd.de

[1] We can alternatively use on-line learning, studied by Amari [1967], Heskes and Kappen [1991], and recently by Barkai et al. [1994] and Solla and Saard [1995].

# References

Amari, S. [1967], *IEEE Trans.*, **EC-16**, 299–307.

Amari, S., Murata, N. [1993], *Neural Computation* 5, 140

Amari, S., Murata, N., Müller, K.-R., Finke, M., Yang, H. [1995], Statistical Theory of Overtraining and Overfitting, Univ. of Tokyo Tech. Report 95-06, submitted

Barkai, N. and Seung, H. S. and Sompolinski, H. [1994], On-line learning of dichotomies, NIPS'94

Finke, M. and Müller, K.-R. [1994] in Proc. of the 1993 Connectionist Models summer school, Mozer, M., Smolensky, P., Touretzky, D.S., Elman, J.L. and Weigend, A.S. (Eds.), Hillsdale, NJ: Erlenbaum Associates, 324

Hassoun, M. H. [1995], Fundamentals of Artificial Neural Networks, MIT Press.

Hecht-Nielsen, R. [1989], Neurocomputing, Addison-Wesley.

Heskes, T. and Kappen, B. [1991], *Physical Review*, **A44**, 2718–2762.

LeCun, Y., Denker, J.S., Solla, S. [1990], Optimal brain damage, NIPS'89

Moody, J. E. [1992], The effective number of parameters: An analysis of generalization and regularization in nonlinear learning systems, NIPS 4

Murata, N., Yoshizawa, S., Amari, S. [1994], *IEEE Trans.*, **NN5**, 865–872.

Müller, K.-R., Finke, M., Murata, N., Schulten, K. and Amari, S. [1995] A numerical study on learning curves in stochastic multilayer feed-forward networks, Univ. of Tokyo Tech. Report METR 95-03 and *Neural Computation* in Press

Poggio, T. and Girosi, F. [1990], *Science*, **247**, 978–982.

Rissanen, J. [1986], *Ann, Statist.*, **14**, 1080–1100.

Rumelhart, D., Hinton, G. E., Williams, R. J. [1986], in PDP, Vol.1, MIT Press.

Saad, D., Solla, S. A. [1995], *PRL*, **74**, 4337 and *Phys. Rev. E*, 52, 4225

Wang, Ch., Venkatesh, S. S., Judd, J. S. [1994], Optimal stopping and effective machine complexity in learning, to appear, (revised and extended version of NIPS'93).